# Agnostic Classification of Markovian Sequences

**Ran El-Yaniv**     **Shai Fine**     **Naftali Tishby**[*]
Institute of Computer Science and Center for Neural Computation
The Hebrew University
Jerusalem 91904, Israel
E-mail: {ranni,fshai,tishby}@cs.huji.ac.il
Category: Algorithms.

## Abstract

Classification of finite sequences without explicit knowledge of their statistical nature is a fundamental problem with many important applications. We propose a new information theoretic approach to this problem which is based on the following ingredients: (i) sequences are similar when they are likely to be generated by the same source; (ii) cross entropies can be estimated via "universal compression"; (iii) Markovian sequences can be asymptotically-optimally merged.

With these ingredients we design a method for the classification of discrete sequences whenever they can be compressed. We introduce the method and illustrate its application for hierarchical clustering of languages and for estimating similarities of protein sequences.

## 1 Introduction

While the relationship between compression (minimal description) and supervised learning is by now well established, no such connection is generally accepted for the unsupervised case. Unsupervised classification is still largely based on ad-hock distance measures with often no explicit statistical justification. This is particularly true for unsupervised classification of sequences of discrete symbols which is encountered in numerous important applications in machine learning and data mining, such as text categorization, biological sequence modeling, and analysis of spike trains.

The emergence of "universal" (i.e. asymptotically distribution independent) se-

---

[*]Corresponding author.

quence compression techniques suggests the existence of "universal" classification methods that make minimal assumptions about the statistical nature of the data. Such techniques are potentially more robust and appropriate for real world applications.

In this paper we introduce a specific method that utilizes the connection between universal compression and unsupervised classification of sequences. Our only underlying assumption is that the sequences can be approximated (in the information theoretic sense) by *some* finite order Markov sources. There are three ingredients to our approach. The first is the assertion that two sequences are statistically similar if they are likely to be independently generated by the same source. This likelihood can then be estimated, given a typical sequence of the most likely joint source, using any good compression method for the sequence samples. The third ingredient is a novel and simple randomized sequence merging algorithm which provably generates a *typical sequence* of the most likely joint source of the sequences, under the above Markovian approximation assumption.

Our similarity measure is also motivated by the known "two sample problem" [Leh59] of estimating the probability that two given samples are taken from the same distribution. In the i.i.d. (Bernoulli) case this problem was thoroughly investigated and the optimal statistical test is given by the sum of the empirical *cross entropies* between the two samples and their *most likely joint source*. We argue that this measure can be extended for arbitrary order Markov sources and use it to construct and sample the most likely joint source.

The similarity measure and the statistical merging algorithm can be naturally combined into classification algorithms for sequences. Here we apply the method to hierarchical clustering of short text segments in 18 European languages and to evaluation of similarities of protein sequences. A complete analysis of the method, with further applications, will be presented elsewhere [EFT97].

## 2   Measuring the statistical similarity of sequences

Estimating the statistical similarity of two individual sequences is traditionally done by training a statistical model for each sequence and then measuring the likelihood of the other sequence by the model. Training a model entails an assumption about the nature of the noise in the data and this is the rational behind most "edit distance" measures, even when the noise model is not explicitly stated.

Estimating the log-likelihood of a sequence-sample over a discrete alphabet $\Sigma$ by a statistical model can be done through the *Cross Entropy* or *Kullback-Leibler Divergence*[CT91] between the sample empirical distribution $p$ and model distribution $q$, defined as:

$$D_{KL}(p||q) = \sum_{\sigma \in \Sigma} p(\sigma) \log \frac{p(\sigma)}{q(\sigma)} . \tag{1}$$

The KL-divergence, however, has some serious practical drawbacks. It is non-symmetric and unbounded unless the model distribution $q$ is absolutely continuous with respect to $p$ (i.e. $q = 0 \Rightarrow p = 0$). The KL-divergence is therefore highly sensitive to low probability events under $q$. Using the "empirical" (sample) distributions for both $p$ and $q$ can result in very unreliable estimates of the true divergences. Essentially, $D_{KL}[p||q]$ measures the asymptotic coding inefficiency when coding the sample $p$ with an optimal code for the model distribution $q$.

The symmetric divergence, i.e. $D(p,q) = D_{KL}[p||q] + D_{KL}[q||p]$, suffers from

similar sensitivity problems and lacks the clear statistical meaning.

## 2.1 The "two sample problem"

Direct Bayesian arguments, or alternately the method of types [CK81], suggest that the probability that there exists one source distribution $\hat{M}$ for two independently drawn samples, $x$ and $y$ [Leh59], is proportional to

$$\int d\mu\left(M\right) \Pr\left(x|M\right) \cdot \Pr\left(y|M\right) = \int d\mu\left(M\right) \cdot 2^{-\left(|x|D_{KL}[p_x||M]+|y|D_{KL}[p_y||M]\right)}, \quad (2)$$

where $d\mu(M)$ is a prior density of all candidate distributions, $p_x$ and $p_y$ are the empirical (sample) distributions, and $|x|$ and $|y|$ are the corresponding sample sizes.

For large enough samples this integral is dominated (for any non-vanishing prior) by the maximal exponent in the integrand, or by *the most likely joint source* of $x$ and $y$, $M_\lambda$, defined as

$$M_\lambda = \arg\min_{M'} \left\{|x|D_{KL}\left(p_x||M'\right) + |y|D_{KL}\left(p_y||M'\right)\right\}. \quad (3)$$

where $0 \le \lambda = |x|/(|x| + |y|) \le 1$ is the sample *mixture ratio*. The convexity of the KL-divergence guarantees that this minimum is unique and is given by

$$M_\lambda = \lambda p_x + (1 - \lambda) p_y,$$

the $\lambda - mixture$ of $p_x$ and $p_y$.

The similarity measure between two samples, $d(x,y)$, naturally follows as the minimal value of the above exponent. That is,

**Definition 1** *The similarity measure, $d(x,y) = \mathcal{D}_\lambda(p_x, p_y)$, of two samples $x$ and $y$, with empirical distributions $p_x$ and $p_y$ respectively, is defined as*

$$d(x,y) = \mathcal{D}_\lambda(p_x, p_y) = \lambda D_{KL}\left(p_x||M_\lambda\right) + (1 - \lambda) D_{KL}\left(p_y||M_\lambda\right) \quad (4)$$

*where $M_\lambda$ is the $\lambda$-mixture of $p_x$ and $p_y$.*

The function $\mathcal{D}_\lambda\left(p, q\right)$ is an extension of the Jensen-Shannon divergence (see e.g. [Lin91]) and satisfies many useful analytic properties, such as symmetry and boundedness on both sides by the $L_1$-norm, in addition to its clear statistical meaning. See [Lin91, EFT97] for a more complete discussion of this measure.

## 2.2 Estimating the $\mathcal{D}_\lambda$ similarity measure

The key component of our classification method is the estimation of $\mathcal{D}_\lambda$ for individual finite sequences, without an explicit model distribution.

Since cross entropies, $D_{KL}$, express code-length differences, they can be estimated using any efficient compression algorithm for the two sequences. The existence of "universal" compression methods, such as the Lempel-Ziv algorithm (see e.g. [CT91]) which are provably asymptotically optimal for any sequence, give us the means for asymptotically optimal estimation of $\mathcal{D}_\lambda$, provided that we can obtain a typical sequence of the most-likely joint source, $M_\lambda$.

We apply an improvement of the method of Ziv and Merhav [ZM93] for the estimation of the two cross-entropies using the Lempel-Ziv algorithm given two sample sequences [BE97]. Notice that our estimation of $\mathcal{D}_\lambda$ is as good as the compression method used, namely, closer to optimal compression yields better estimation of the similarity measure.

It remains to show how a typical sequence of the most-likely joint source can be generated.

## 3   Joint Sources of Markovian Sequences

In this section we first explicitly generalize the notion of the joint statistical source to finite order Markov probability measures. We identify the joint source of Markovian sequences and show how to construct a typical random sample of this source.

More precisely, let $x$ and $y$ be two sequences generated by Markov processes with distributions $P$ and $Q$, respectively. We present a novel algorithm for the merging the two sequences, by generating a typical sequence of an approximation to the most likely joint source of $x$ and $y$. The algorithm does not require the parameters of the true sources $P$ and $Q$ and the computation of the sequence is done directly from the sequence samples $x$ and $y$.

As before, $\Sigma$ denotes a finite alphabet and $P$ and $Q$ denote two ergodic Markov sources over $\Sigma$ of orders $K_P$ and $K_Q$, respectively. By equation 3, the $\lambda$-*mixture joint source* $M_\lambda$ of $P$ and $Q$ is $M_\lambda = \arg\min_{M'} \lambda D_{KL}(P||M') + (1-\lambda)D_{KL}(Q||M')$, where for sequences $D_{KL}(P||M) = \limsup_{n\to\infty} \frac{1}{n}\sum_{x\in\Sigma^n} P(x)\log\frac{P(x)}{M(x)}$. The following theorem identifies the joint source of $P$ and $Q$.

**Theorem 1** *The unique $\lambda$-mixture joint source $M_\lambda$ of $P$ and $Q$, of order $K = \max\{K_P, K_Q\}$, is given by the following conditional distribution. For each $s \in \Sigma^K, a \in \Sigma$,*

$$M_\lambda(a|s) = \frac{\lambda P(s)}{\lambda P(s) + (1-\lambda)Q(s)}P(a|s) + \frac{(1-\lambda)Q(s)}{\lambda P(s) + (1-\lambda)Q(s)}Q(a|s) \ .$$

This distribution can be naturally extended to $n$ sources with priors $\lambda_1, \ldots, \lambda_n$.

### 3.1   The "sequence merging" algorithm

The above theorem can be easily translated into an algorithm. Figure 1 describes a randomized algorithm that generates from the given sequences $x$ and $y$, an asymptotically typical sequence $z$ of the most likely joint source, as defined by Theorem 1, of $P$ and $Q$.

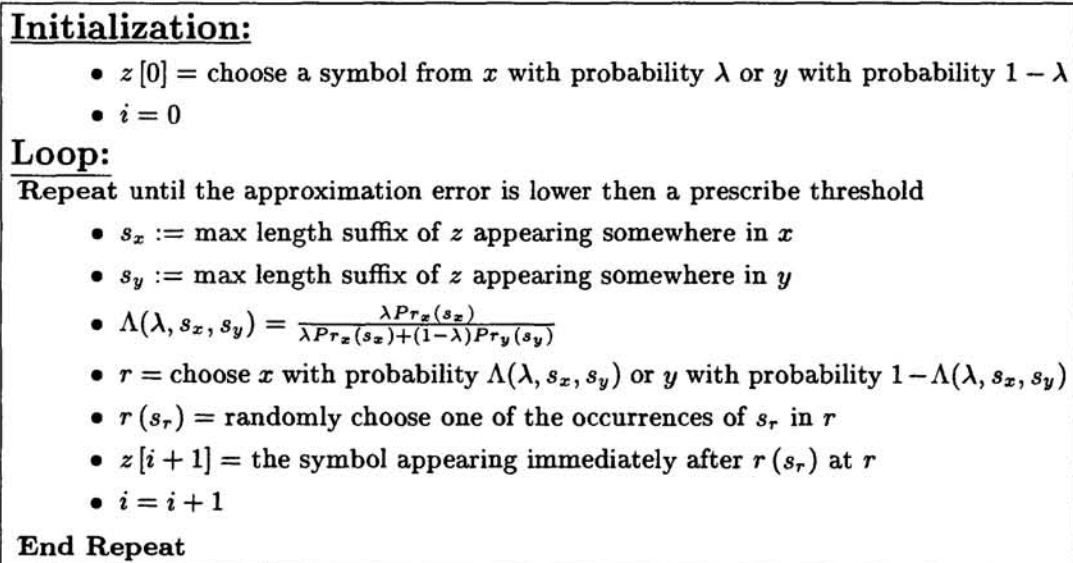

Figure 1: The most-likely joint source algorithm

Notice that the algorithm is completely unparameterized, even the sequence alphabets, which may differ from one sequence to another, are not explicitly needed. The algorithm can be efficiently implemented by pre-preparing suffix trees for the given sequences, and the merging algorithm is naturally generalizable to any number of sequences.

## 4  Applications

There are several possible applications of our sequence merging algorithm and similarity measure. Here we focus on three possible applications: the source merging problem, estimation of sequence similarity, and bottom-up sequence-classification. These algorithms are different from most existing approaches because they rely only on the sequenced data, similar to universal compression, without explicit modeling assumptions. Further details, analysis, and applications of the method will be presented elsewhere [EFT97].

### 4.1  Merging and synthesis of sequences

An immediate application of the source merging algorithm is for synthesis of typical sequences of the joint source from some given data sequences, *without any access to an explicit model of the source.*

To illustrate this point consider the sequence in Figure 2. This sequence was randomly generated, *character by character*, from two natural excerpts: a 47,655-character string from Dickens' Tale of Two Cities, and a 59,097-character string from Twain's The King and the Pauper.

```
        Do your way to her breast, and sent a treason's sword- and not empty.
    "I am particularly and when the stepped of his own commits place.  No; yes,
    of course, and he passed behind that by turns ascended upon him, and my bone
    to touch it, less to say:  'Remove thought, every one!  Guards!  In minessa?"
    The books third time.  There was but pastened her unave misg his ruined head
    than they had known to keep his saw whether think" The feet our grace he
    called offer information?
```

[Twickens, 1997]

Figure 2: A typical excerpt of random text generated by the "joint source" of Dickens and Twain.

### 4.2  Pairwise similarity of proteins

The joint source algorithm, combined with the new similarity measure, provide natural means for computing the similarity of sequences over any alphabet. In this section we illustrate this application[1] for the important case of protein sequences (sequences over the set of the 20 amino-acids).

From a database of all known proteins we selected 6 different families and within each family we randomly chose 10 proteins. The families chosen are: *Chaperonin, MHC1, Cytochrome, Kinase, Globin Alpha and Globin Beta.* Our pairwise distances between all 60 proteins were computed using our agnostic algorithm and are depicted in the 60x60 matrix of Figure 3. As can be seen, the algorithm succeeds to

identify the families (the success with the Kinase and Cytochrome families is more limited).

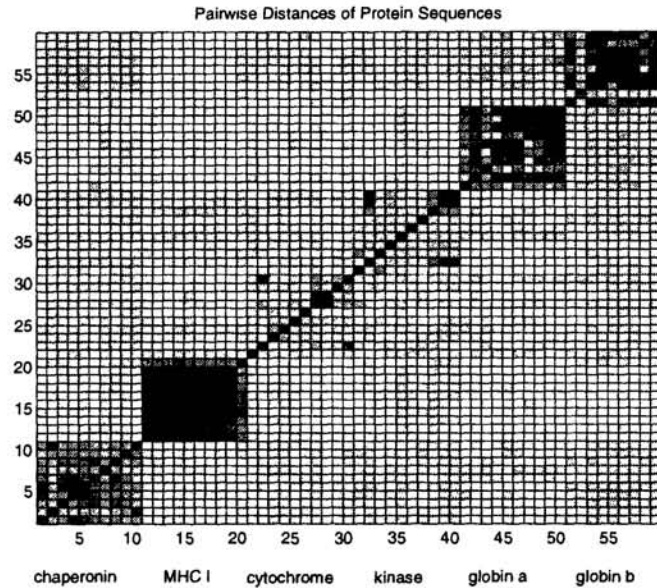

Figure 3: A 60x60 symmetric matrix representing the pairwise distances, as computed by our agnostic algorithm, between 60 proteins, each consecutive 10 belong to a different family. Darker gray represent higher similarity.

In another experiment we considered all the 200 proteins of the Kinase family and computed the pairwise distances of these proteins using the agnostic algorithm. For comparison we computed the pairwise similarities of these sequences using the widely used Smith-Waterman algorithm (see e.g. [HH92]).[2] The resulting agnostic similarities, computed with no biological information whatsoever, are very similar to the Smith-Waterman similarities.[3] Furthermore, our agnostic measure discovered some biological similarities not detected by the Smith-Waterman method.

## 4.3 Agnostic classification of languages

The sample of the joint source of two sequences can be considered as their "average" or "centroid", capturing a mixture of their statistics. Averaging and measuring distance between objects are sufficient for most standard clustering algorithms such as bottom-up greedy clustering, vector quantization (VQ), and clustering by deterministic annealing. Thus, our merging method and similarity measure can be directly applied for the classification of finite sequences via standard clustering algorithms.

To illustrate the power of this new sequence clustering method we give the result of a rudimentary linguistic experiment using a greedy bottom-up (conglomerative) clustering of short excerpts (1500 characters) from eighteen languages. Specifically, we took sixteen random excerpts from the following Porto-Indo-European languages: *Afrikaans, Catalan, Danish, Dutch, English, Flemish, French, German, Italian, Latin, Norwegian, Polish, Portuguese, Spanish, Swedish* and *Welsh*, together with

two artificial languages: *Esperanto* and *Klingon*[4].

The resulting hierarchical classification tree is depicted in Figure 4. This entirely unsupervised method, when applied to these short random excerpts, clearly agrees with the "standard" philologic tree of these languages, both in terms of the grouping and the levels of similarity (depth of the split) of the languages (the Polish-Welsh "similarity" is probably due to the specific transcription used).

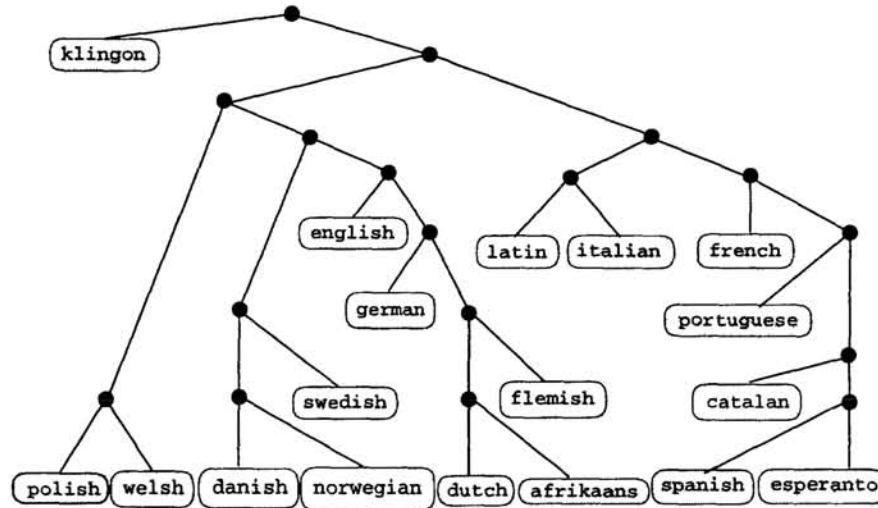

Figure 4: Agnostic bottom-up greedy clustering of eighteen languages

## Acknowledgments

We sincerely thank Ran Bachrach and Golan Yona for helpful discussions. We also thank Sageev Oore for many useful comments.

## Footnotes

[1]The protein results presented here are part of an ongoing work with G. Yona and E. Ben-Sasson.

[2]we applied the Smith-Waterman for computing local-alignment costs using the state-of-the-art *blosum62* biological cost matrix.

[3]These results are not given here due to space limitations and will be discussed elsewhere.

[4]Klingon is a synthetic language that was invented for the Star-Trek TV series.

## References

[BE97]   R. Bachrach and R. El-Yaniv, An Improved Measure of Relative Entropy Between Individual Sequences, unpublished manuscript.

[CK81]   I. Csiszár and J. Krörner. Information Theory: Coding Theorems for Discrete Memoryless Systems, Academic Press, New-York 1981.

[CT91]   T. M. Cover and J. A. Thomas. Elements of Information Theory, John Wiley & Sons, New-York 1991.

[EFT97]  R. El-Yaniv, S. Fine and N. Tishby. Classifying Markovian Sources, in preparations, 1997.

[HH92]   S. Henikoff and J. G. Henikoff (1992). Amino acid substitution matrices from protein blocks. *Proc. Natl. Acad. Sci. USA* **89**, 10915-10919.

[Leh59]  E. L. Lehmann. Testing Statistical Hypotheses, John Wiley & Sons, New-York 1959.

[Lin91]  J. Lin, 1991. Divergence measures based on the Shannon entropy. *IEEE Transactions on Information Theory*, 37(1):145–151.

[ZM93]   J. Ziv and N. Merhav, 1993. A Measure of Relative Entropy Between Individual Sequences with Application to Universal Classification, *IEEE Transactions on Information Theory*, 39(4).

